# Warped Gaussian Processes

**Edward Snelson**[*]        **Carl Edward Rasmussen**[†]        **Zoubin Ghahramani**[*]

[*]Gatsby Computational Neuroscience Unit
University College London
17 Queen Square, London WC1N 3AR, UK
{snelson,zoubin}@gatsby.ucl.ac.uk

[†]Max Planck Institute for Biological Cybernetics
Spemann Straße 38, 72076 Tübingen, Germany
carl@tuebingen.mpg.de

## Abstract

We generalise the Gaussian process (GP) framework for regression by learning a nonlinear transformation of the GP outputs. This allows for non-Gaussian processes and non-Gaussian noise. The learning algorithm chooses a nonlinear transformation such that transformed data is well-modelled by a GP. This can be seen as including a preprocessing transformation as an integral part of the probabilistic modelling problem, rather than as an ad-hoc step. We demonstrate on several real regression problems that learning the transformation can lead to significantly better performance than using a regular GP, or a GP with a fixed transformation.

## 1   Introduction

A Gaussian process (GP) is an extremely concise and simple way of placing a prior on functions. Once this is done, GPs can be used as the basis for nonlinear nonparametric regression and classification, showing excellent performance on a wide variety of datasets [1, 2, 3]. Importantly they allow full Bayesian predictive distributions to be obtained, rather than merely point predictions.

However, in their simplest form GPs are limited by the nature of their simplicity: they assume the target data to be distributed as a multivariate Gaussian, with Gaussian noise on the individual points. This simplicity enables predictions to be made easily using matrix manipulations, and of course the predictive distributions are Gaussian also.

Often it is unreasonable to assume that, in the form the data is obtained, the noise will be Gaussian, and the data well modelled as a GP. For example, the observations may be positive quantities varying over many orders of magnitude, where it makes little sense to model these quantities directly assuming homoscedastic Gaussian noise. In these situations it is standard practice in the statistics literature to take the log of the data. Then modelling proceeds assuming that this transformed data has Gaussian noise and will be better modelled by the GP. The log is just one particular transformation that could be done; there is a con-

tinuum of transformations that could be applied to the observation space to bring the data into a form well modelled by a GP. Making such a transformation should really be a full part of the probabilistic modelling; it seems strange to first make an ad-hoc transformation, and then use a principled Bayesian probabilistic model.

In this paper we show how such a transformation or 'warping' of the observation space can be made entirely automatically, fully encompassed into the probabilistic framework of the GP. The warped GP makes a transformation from a latent space to the observation, such that the data is best modelled by a GP in the latent space. It can also be viewed as a generalisation of the GP, since in observation space it is a non-Gaussian process, with non-Gaussian and asymmetric noise in general. It is not however just a GP with a non-Gaussian noise model; see section 6 for further discussion.

For an excellent review of Gaussian processes for regression and classification see [4]. We follow the notation there throughout this paper and present a brief summary of GP regression in section 2. We show in sections 4 and 5, with both toy and real data, that the warped GP can significantly improve predictive performance over a variety of measures, especially with regard to the whole predictive distribution, rather than just a single point prediction such as the mean or median. The transformation found also gives insight into the properties of the data.

## 2 Nonlinear regression with Gaussian processes

Suppose we are given a dataset $\mathcal{D}$, consisting of $N$ pairs of input vectors $\mathbf{X}_N \equiv \{\mathbf{x}^{(n)}\}_{n=1}^N$ and real-valued targets $\mathbf{t}_N \equiv \{t_n\}_{n=1}^N$. We wish to predict the value of an observation $t_{N+1}$ given a new input vector $\mathbf{x}^{(N+1)}$, or rather the distribution $P(t_{N+1}|\mathbf{x}^{(N+1)}, \mathcal{D})$. We assume there is an underlying function $y(\mathbf{x})$ which we are trying to model, and that the observations lie noisily around this. A GP places a prior directly on the space of functions by assuming that any finite selection of points $\mathbf{X}_N$ gives rise to a multivariate Gaussian distribution over the corresponding function values $\mathbf{y}_N$. The covariance between the function value of $y$ at two points $\mathbf{x}$ and $\mathbf{x}'$ is modelled with a covariance function $C(\mathbf{x}, \mathbf{x}')$, which is usually assumed to have some simple parametric form. If the noise model is taken to be Gaussian, then the distribution over observations $\mathbf{t}_N$ is also Gaussian with the entries of the covariance matrix $\mathbf{C}$ given by

$$C_{mn} = C(\mathbf{x}^{(m)}, \mathbf{x}^{(n)}; \mathbf{\Theta}) + \delta_{mn} g(\mathbf{x}^{(n)}; \mathbf{\Theta}) \,, \tag{1}$$

where $\mathbf{\Theta}$ parameterises the covariance function, $g$ is the noise model, and $\delta_{mn}$ is the Kronecker delta function.

Often the noise model is taken to be input-independent, and the covariance function is taken to be a Gaussian function of the difference in the input vectors (a *stationary* covariance function), although many other possibilities exist, see e.g. [5] for GPs with input dependent noise. In this paper we consider only this popular choice, in which case the entries in the covariance matrix are given by

$$C_{mn} = v_1 \exp\left[ -\frac{1}{2} \sum_{d=1}^{D} \left( \frac{x_d^{(m)} - x_d^{(n)}}{r_d} \right)^2 \right] + v_0 \delta_{mn} \,. \tag{2}$$

Here $r_d$ is a width parameter expressing the scale over which typical functions vary in the $d$th dimension, $v_1$ is a size parameter expressing the typical size of the overall process in $y$-space, $v_0$ is the noise variance of the observations, and $\mathbf{\Theta} = \{v_0, v_1, r_1, \ldots, r_D\}$.

It is simple to show that the predictive distribution for a new point given the observed data, $P(t_{N+1}|\mathbf{t}_N, \mathbf{X}_{N+1})$, is Gaussian. The calculation of the mean and variance of this

distribution involves doing a matrix inversion of the covariance matrix $\mathbf{C}_N$ of the training inputs, which using standard exact methods incurs a computational cost of order $N^3$.

Learning, or 'training', in a GP is usually achieved by finding a local maximum in the likelihood using conjugate gradient methods with respect to the hyperparameters $\mathbf{\Theta}$ of the covariance matrix. The negative log likelihood is given by

$$\mathcal{L} = -\log P(\mathbf{t}_N | \mathbf{X}_N, \mathbf{\Theta}) = \frac{1}{2} \log \det \mathbf{C}_N + \frac{1}{2} \mathbf{t}_N^\top \mathbf{C}_N^{-1} \mathbf{t}_N + \frac{N}{2} \log 2\pi . \qquad (3)$$

Once again, the evaluation of $\mathcal{L}$, and its gradients with respect to $\mathbf{\Theta}$, involve computing the inverse covariance matrix, incurring an order $N^3$ cost. Rather than finding a ML estimate $\mathbf{\Theta}_{\mathrm{ML}}$, a prior over $\mathbf{\Theta}$ can be included to find a MAP estimate $\mathbf{\Theta}_{\mathrm{MAP}}$, or even better $\mathbf{\Theta}$ can be numerically integrated out when computing $P(t_{N+1} | \mathbf{x}^{(N+1)}, \mathcal{D})$ using for example hybrid Monte Carlo methods [2, 6].

## 3 Warping the observation space

In this section we present a method of warping the observation space through a nonlinear monotonic function to a latent space, whilst retaining the full probabilistic framework to enable learning and prediction to take place consistently. Let us consider a vector of latent targets $\mathbf{z}_N$ and suppose that this vector *is* modelled by a GP,

$$-\log P(\mathbf{z}_N | \mathbf{X}_N, \mathbf{\Theta}) = \frac{1}{2} \log \det \mathbf{C}_N + \frac{1}{2} \mathbf{z}_N^\top \mathbf{C}_N^{-1} \mathbf{z}_N + \frac{N}{2} \log 2\pi . \qquad (4)$$

Now we make a transformation from the true observation space to the latent space by mapping each observation through the same monotonic function $f$,

$$z_n = f(t_n; \mathbf{\Psi}) \qquad \forall n , \qquad (5)$$

where $\mathbf{\Psi}$ parameterises the transformation. We require $f$ to be monotonic and mapping on to the whole of the real line; otherwise probability measure will not be conserved in the transformation, and we will not induce a valid distribution over the targets $\mathbf{t}_N$. Including the Jacobian term that takes the transformation into account, the negative log likelihood, $-\log P(\mathbf{t}_N | \mathbf{X}_N, \mathbf{\Theta}, \mathbf{\Psi})$, now becomes:

$$\mathcal{L} = \frac{1}{2} \log \det \mathbf{C}_N + \frac{1}{2} f(\mathbf{t}_N)^\top \mathbf{C}_N^{-1} f(\mathbf{t}_N) - \sum_{n=1}^{N} \log \left. \frac{\partial f(t)}{\partial t} \right|_{t_n} + \frac{N}{2} \log 2\pi . \qquad (6)$$

### 3.1 Training the warped GP

Learning in this extended model is achieved by simply taking derivatives of the negative log likelihood function (6) with respect to both $\mathbf{\Theta}$ and $\mathbf{\Psi}$ parameter vectors, and using a conjugate gradient method to compute ML parameter values. In this way the form of both the covariance matrix and the nonlinear transformation are learnt simultaneously under the same probabilistic framework. Since the computational limiter to a GP is inverting the covariance matrix, adding a few extra parameters into the likelihood is not really costing us anything. All we require is that the derivatives of $f$ are easy to compute (both with respect to $t$ and $\mathbf{\Psi}$), and that we don't introduce so many extra parameters that we have problems with over-fitting. Of course a prior over both $\mathbf{\Theta}$ and $\mathbf{\Psi}$ may be included to compute a MAP estimate, or in fact the parameters integrated out using a hybrid Monte Carlo method.

### 3.2 Predictions with the warped GP

For a particular setting of the covariance function hyperparameters $\mathbf{\Theta}$ (for example $\mathbf{\Theta}_{\mathrm{ML}}$ or $\mathbf{\Theta}_{\mathrm{MAP}}$), in *latent* variable space the predictive distribution at a new point is just as for a

regular GP: a Gaussian whose mean and variance are calculated as mentioned in section 2;

$$P(z_{N+1}|\mathbf{x}^{(N+1)}, \mathcal{D}, \boldsymbol{\Theta}) = \mathcal{N}\left(\hat{z}_{N+1}(\boldsymbol{\Theta}), \sigma_{N+1}^2(\boldsymbol{\Theta})\right) . \tag{7}$$

To find the distribution in the observation space we pass that Gaussian through the nonlinear warping function, giving

$$P(t_{N+1}|\mathbf{x}^{(N+1)}, \mathcal{D}, \boldsymbol{\Theta}, \boldsymbol{\Psi}) = \frac{f'(t_{N+1})}{\sqrt{2\pi\sigma_{N+1}^2}} \exp\left[-\frac{1}{2}\left(\frac{f(t_{N+1}) - \hat{z}_{N+1}}{\sigma_{N+1}}\right)^2\right] . \tag{8}$$

The shape of this distribution depends on the form of the warping function $f$, but in general it may be asymmetric and multimodal.

If we require a point prediction to be made, rather than the whole distribution over $t_{N+1}$, then the value we will predict depends on our loss function. If our loss function is absolute error, then the median of the distribution should be predicted, whereas if our loss function is squared error, then it is the mean of the distribution. For a standard GP where the predictive distribution is Gaussian, the median and mean lie at the same point. For the warped GP in general they are at different points. The median is particularly easy to calculate:

$$t_{N+1}^{\text{med}} = f^{-1}(\hat{z}_{N+1}) . \tag{9}$$

Notice we need to compute the inverse warping function. In general we are unlikely to have an analytical form for $f^{-1}$, because we have parameterised the function in the opposite direction. However since we have access to derivatives of $f$, a few iterations of Newton-Raphson with a good enough starting point is enough.

It is often useful to give an indication of the shape and range of the distribution by giving the positions of various 'percentiles'. For example we may want to know the positions of '$2\sigma$' either side of the median so that we can say that approximately 95% of the density lies between these bounds. These points in observation space are calculated in exactly the same way as the median - simply pass the values through the inverse function:

$$t_{N+1}^{\text{med}\pm 2\sigma} = f^{-1}(\hat{z}_{N+1} \pm 2\sigma_{N+1}) . \tag{10}$$

To calculate the mean, we need to integrate $t_{N+1}$ over the density (8). Rewriting this integral back in latent space we get

$$\mathcal{E}(t_{N+1}) = \int dz f^{-1}(z)\mathcal{N}_z(\hat{z}_{N+1}, \sigma_{N+1}^2) = \mathcal{E}(f^{-1}) . \tag{11}$$

This is a simple one dimensional integral under a Gaussian density, so Gauss-Hermite quadrature may be used to accurately compute it with a weighted sum of a small number of evaluations of the inverse function $f^{-1}$ at appropriate places.

### 3.3 Choosing a monotonic warping function

We wish to design a warping function that will allow for complex transformations, but we must constrain the function to be monotonic. There are various ways to do this, an obvious one being a neural-net style sum of $\tanh$ functions,

$$f(t; \boldsymbol{\Psi}) = \sum_{i=1}^{I} a_i \tanh\left(b_i(t + c_i)\right) \qquad a_i, b_i \geq 0 \quad \forall i , \tag{12}$$

where $\boldsymbol{\Psi} = \{\mathbf{a}, \mathbf{b}, \mathbf{c}\}$. This produces a series of smooth steps, with $\mathbf{a}$ controlling the size of the steps, $\mathbf{b}$ controlling their steepness, and $\mathbf{c}$ their position. Of course the number of

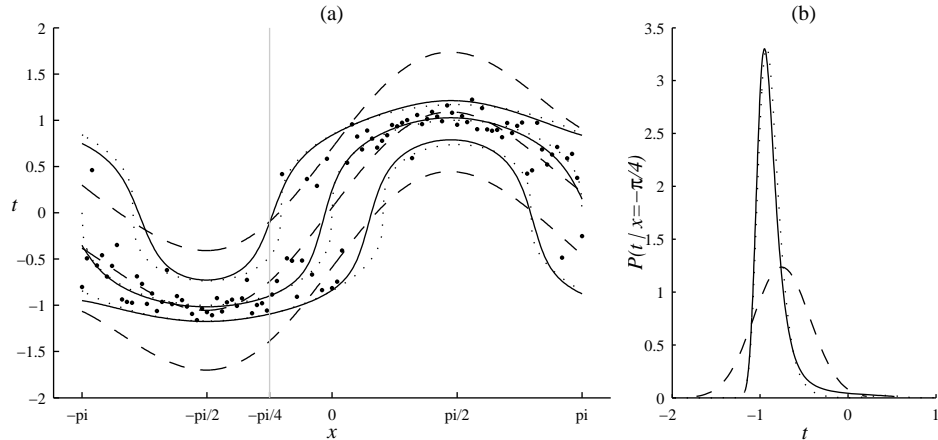

Figure 1: A 1D regression task. The dotted lines show the true generating distribution, the dashed lines show a GP's predictions, and the solid lines show the warped GP's predictions. (a) The triplets of lines represent the median, and $2\sigma$ percentiles in each case. (b) Predictive probability densities at $x = -\pi/4$; i.e. a cross section through (a) at the solid grey line

steps $I$ needs to be set, and that will depend on how complex a function one wants. The derivatives of this function with respect to either $t$, or the warping parameters $\boldsymbol{\Psi}$, are easy to compute. In the same spirit, sums of error functions, or sums of logistic functions, would produce a similar series of steps, and so these could be used instead.

The problem with using (12) as it stands is that it is bounded; the inverse function $f^{-1}(z)$ does not exist for values of $z$ outside the range of these bounds. As explained earlier, this will not lead to a proper density in $t$ space, because the density in $z$ space is Gaussian, which covers the whole of the real line. We can fix this up by using instead:

$$f(t; \boldsymbol{\Psi}) = t + \sum_{i=1}^{I} a_i \tanh\left(b_i(t + c_i)\right) \qquad a_i, b_i \geq 0 \quad \forall i .  \tag{13}$$

which has linear trends away from the $\tanh$ steps. In doing so, we have restricted ourselves to only making warping functions with $f' \geq 1$, but because the size of the covariance function $v_1$ is free to vary, the *effective* gradient can be made arbitrarily small by simply making the range of the data in the latent space arbitrarily big.

A more flexible system of linear trends may be made by including, in addition to the neural-net style function (12), some functions of the form $\frac{1}{\beta} \log\left[e^{\beta m_1(t-d)} + e^{\beta m_2(t-d)}\right]$, where $m_1, m_2 \geq 0$. This function effectively splices two straight lines of gradients $m_1$ and $m_2$ smoothly together with a 'curvature' parameter $\beta$, and at position $d$. The sign of $\beta$ determines whether the join is convex or concave.

## 4   A simple 1D regression task

A simple 1D regression task was created to show a situation where the warped GP should, and does, perform significantly better than the standard GP. 101 points, regularly spaced from $-\pi$ to $\pi$ on the $x$ axis, were generated with Gaussian noise about a sine function. These points were then warped through the function $t = z^{1/3}$, to arrive at the dataset $\mathbf{t}$ which is shown as the dots in Figure 1(a).

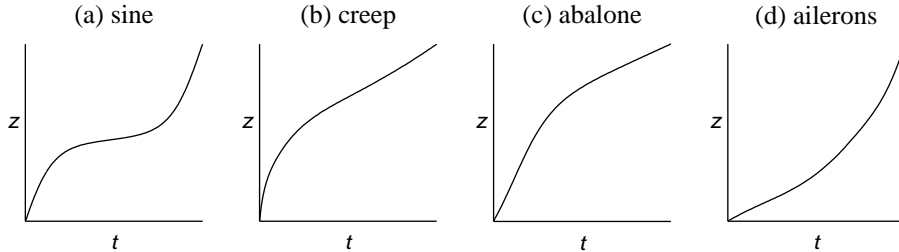

Figure 2: Warping functions learnt for the four regression tasks carried out in this paper. Each plot is made over the range of the observation data, from $t_{\min}$ to $t_{\max}$.

A GP and a warped GP were trained independently on this dataset using a conjugate gradient minimisation procedure and randomly initialised parameters, to obtain maximum likelihood parameters. For the warped GP, the warping function (13) was used with just two $\tanh$ functions. For both models the covariance matrix (2) was used. Hybrid Monte Carlo was also implemented to integrate over all the parameters, or just the warping parameters (much faster since no matrix inversion is required with each step), but with this dataset (and the real datasets of section 5) no significant differences were found from ML.

Predictions from the GP and warped GP were made, using the ML parameters, for 401 points regularly spaced over the range of $x$. The predictions made were the median and $2\sigma$ percentiles in each case, and these are plotted as triplets of lines on Figure 1(a). The predictions from the warped GP are found to be much closer to the true generating distribution than the standard GP, especially with regard to the $2\sigma$ lines. The mean line was also computed, and found to lie close, but slightly skewed, from the median line.

Figure 1(b) emphasises the point that the warped GP finds the shape of the whole predictive *distribution* much better, not just the median or mean. In this plot, one particular point on the $x$ axis is chosen, $x = -\pi/4$, and the predictive densities from the GP and warped GP are plotted alongside the true density (which can be written down analytically). Note that the standard GP must necessarily predict a symmetrical Gaussian density, even when the density from which the points are generated is highly asymmetrical, as in this case.

Figure 2(a) shows the warping function learnt for this regression task. The $\tanh$ functions have adjusted themselves so that they mimic a $t^3$ nonlinearity over the range of the observation space, thus inverting the $z^{1/3}$ transformation imposed when generating the data.

## 5 Results for some real datasets

It is not surprising that the method works well on the toy dataset of section 4 since it was generated from a known nonlinear warping of a smooth function with Gaussian noise. To demonstrate that nonlinear transformations also help on real data sets we have run the warped GP comparing its predictions to an ordinary GP on three regression problems. These datasets are summarised in the following table which shows the range of the targets $(t_{\min}, t_{\max})$, the number of input dimensions $(D)$, and the size of the training and test sets $(N_{\text{train}}, N_{\text{test}})$ that we used.

| Dataset | $D$ | $t_{\min}$ | $t_{\max}$ | $N_{\text{train}}$ | $N_{\text{test}}$ |
|---|---|---|---|---|---|
| creep | 30 | 18 MPa | 530 MPa | 800 | 1266 |
| abalone | 8 | 1 yr | 29 yrs | 1000 | 3177 |
| ailerons | 40 | $-3.0 \times 10^{-3}$ | $-3.5 \times 10^{-4}$ | 1000 | 6154 |

| Dataset | Model | Absolute error | Squared error | $-\log P(t)$ |
|---|---|---|---|---|
| creep | GP | 16.4 | 654 | 4.46 |
| | GP + log | 15.6 | 587 | 4.24 |
| | warped GP | 15.0 | 554 | 4.19 |
| abalone | GP | 1.53 | 4.79 | 2.19 |
| | GP + log | 1.48 | 4.62 | 2.01 |
| | warped GP | 1.47 | 4.63 | 1.96 |
| ailerons | GP | $1.23 \times 10^{-4}$ | $3.05 \times 10^{-8}$ | -7.31 |
| | warped GP | $1.18 \times 10^{-4}$ | $2.72 \times 10^{-8}$ | -7.45 |

Table 1: Results of testing the GP, warped GP, and GP with $\log$ transform, on three real datasets. The units for absolute error and squared error are as for the original data.

The dataset creep is a materials science set, with the objective to predict creep rupture stress (in MPa) for steel given chemical composition and other inputs [7, 8]. With abalone the aim is to predict the the age of abalone from various physical inputs [9]. ailerons is a simulated control problem, with the aim to predict the control action on the ailerons of an F16 aircraft [10, 11].

For datasets creep and abalone, which consist of positive observations only, standard practice may be to model the $\log$ of the data with a GP. So for these datasets we have compared three models: a GP directly on the data, a GP on the fixed log-transformed data, and the warped GP directly on the data. The predictive points and densities were always compared in the original data space, accounting for the Jacobian of both the $\log$ and the warped transforms. The models were run as in the 1D task: ML parameter estimates only, covariance matrix (2), and warping function (13) with three $\tanh$ functions.

The results we obtain for the three datasets are shown in Table 1. We show three measures of performance over independent test sets: mean absolute error, mean squared error, and the mean negative $\log$ predictive density evaluated at the test points. This final measure was included to give some idea of how well the model predicts the entire density, not just point predictions.

On these three sets, the warped GP always performs significantly better than the standard GP. For creep and abalone, the fixed $\log$ transform clearly works well too, but particularly in the case of creep, the warped GP learns a better transformation. Figure 2 shows the warping functions learnt, and indeed 2(b) and 2(c) are clearly $\log$-like in character. On the other hand 2(d), for the ailerons set, is exponential-like. This shows the warped GP is able to flexibly handle these different types of datasets. The shapes of the learnt warping functions were also found to be very robust to random initialisation of the parameters. Finally, the warped GP also makes a better job of predicting the distributions, as shown by the difference in values of the negative $\log$ density.

## 6 Conclusions, extensions, and related work

We have shown that the warped GP is a useful extension to the standard GP for regression, capable of finding extra structure in the data through the transformations it learns. From another viewpoint, it allows standard preprocessing transforms, such as $\log$, to be discovered automatically and improved on, rather than be applied in an ad-hoc manner. We have demonstrated an improvement in performance over the regular GP on several datasets.

Of course some datasets are well modelled by a GP already, and applying the warped GP model simply results in a linear "warping" function. It has also been found that datasets that have been censored, i.e. many observations at the edge of the range lie on a single point,

cause the warped GP problems. The warping function attempts to model the censoring by pushing those points far away from the rest of the data, and it suffers in performance especially for ML learning. To deal with this properly a censorship model is required.

As a further extension, one might consider warping the input space in some nonlinear fashion. In the context of geostatistics this has actually been dealt with by O'Hagan [12], where a transformation is made from an input space which can have non-stationary and non-isotropic covariance structure, to a latent space in which the usual conditions of stationarity and isotropy hold.

Gaussian process classifiers can also be thought of as warping the outputs of a GP, through a mapping onto the $(0, 1)$ probability interval. However, the observations in classification are discrete, not points in this warped continuous space. Therefore the likelihood is different. Diggle et al. [13] consider various other fixed nonlinear transformations of GP outputs.

It should be emphasised that the presented method can be beneficial in situations where the noise variance depends on the output value. Gaussian processes where the noise variance depends on the *inputs* have been examined by e.g. [5]. Forms of non-Gaussianity which do not directly depend on the output values (such as heavy tailed noise) are also not captured by the method proposed here. We propose that the current method should be used in conjunction with methods targeted directly at these other issues. The force of the method it that it is powerful, yet very easy and computationally cheap to apply.

**Acknowledgements**. Many thanks to David MacKay for useful discussions, suggestions of warping functions and datasets to try. CER was supported by the German Research Council (DFG) through grant RA 1030/1.

## References

[1] C. K. I. Williams and C. E. Rasmussen. Gaussian processes for regression. In D. S. Touretzky, M. C. Mozer, and M. E. Hasselmo., editors, *Advances in Neural Information Processing Systems 8*. MIT Press, 1996.

[2] C. E. Rasmussen. *Evaluation of Gaussian Processes and Other Methods for Non-Linear Regression*. PhD thesis, University of Toronto, 1996.

[3] M. N. Gibbs. *Bayesian Gaussian Processes for Regression and Classification*. PhD thesis, Cambridge University, 1997.

[4] D. J. C. MacKay. Introduction to Gaussian processes. In C. M. Bishop, editor, *Neural Networks and Machine Learning*, NATO ASI Series, pages 133–166. Kluwer Academic Press, 1998.

[5] Paul W. Goldberg, Christopher K. I. Williams, and Christopher M. Bishop. Regression with input-dependent noise: A gaussian process treatment. In *Advances in Neural Information Processing Systems 10*. MIT Press, 1998.

[6] Radford M.Neal. Monte Carlo implementation of Gaussian process models for Bayesian regression and classification. Technical Report 9702, University of Toronto, 1997.

[7] Materials algorithms project (MAP) program and data library. `http://www.msm.cam.ac.uk/map/entry.html`.

[8] D. Cole, C. Martin-Moran, A. G. Sheard, H. K. D. H. Bhadeshia, and D. J. C. MacKay. Modelling creep rupture strength of ferritic steel welds. *Science and Technology of Welding and Joining*, 5:81–90, 2000.

[9] C. L. Blake and C. J. Merz. UCI repository of machine learning databases, 1998. `http://www.ics.uci.edu/~mlearn/MLRepository.html`.

[10] L. Torgo. `http://www.liacc.up.pt/~ltorgo/Regression/`.

[11] R. Camacho. *Inducing models of human control skills*. PhD thesis, University of Porto, 2000.

[12] A. O'Hagan and A. M. Schmidt. Bayesian inference for nonstationary spatial covariance structure via spatial deformations. Technical Report 498/00, University of Sheffield, 2000.

[13] P. J. Diggle, J. A. Tawn, and R. A. Moyeed. Model-based geostatistics. *Applied Statistics*, 1998.